# Optimization Principles for the Neural Code

**Michael DeWeese**
Sloan Center, Salk Institute
La Jolla, CA 92037
deweese@salk.edu

## Abstract

Recent experiments show that the neural codes at work in a wide range of creatures share some common features. At first sight, these observations seem unrelated. However, we show that these features arise naturally in a linear filtered threshold crossing (LFTC) model when we set the threshold to maximize the transmitted information. This maximization process requires neural adaptation to not only the DC signal level, as in conventional light and dark adaptation, but also to the statistical structure of the signal and noise distributions. We also present a new approach for calculating the mutual information between a neuron's output spike train and any aspect of its input signal which does not require reconstruction of the input signal. This formulation is valid provided the correlations in the spike train are small, and we provide a procedure for checking this assumption. This paper is based on joint work (DeWeese [1], 1995). Preliminary results from the LFTC model appeared in a previous proceedings (DeWeese [2], 1995), and the conclusions we reached at that time have been reaffirmed by further analysis of the model.

## 1 Introduction

Most sensory receptor cells produce analog voltages and currents which are smoothly related to analog signals in the outside world. Before being transmitted to the brain, however, these signals are encoded in sequences of identical pulses called action potentials or spikes. We would like to know if there is a universal principle at work in the choice of these coding strategies. The existence of such a potentially powerful theoretical tool in biology is an appealing notion, but it may not turn out to be useful. Perhaps the function of biological systems is best seen as a complicated compromise among constraints imposed by the properties of biological materials, the need to build the system according to a simple set of development rules, and

the fact that current systems must arise from their ancestors by evolution through random change and selection. In this view, biology is history, and the search for principles (except for evolution itself) is likely to be futile. Obviously, we hope that this view is wrong, and that at least some of biology is understandable in terms of the same sort of universal principles that have emerged in the physics of the inanimate world.

Adrian noticed in the 1920's that every peripheral neuron he checked produced discrete, identical pulses no matter what input he administered (Adrian, 1928). From the work of Hodgkin and Huxley we know that these pulses are stable non-linear waves which emerge from the non-linear dynamics describing the electrical properties of the nerve cell membrane These dynamics in turn derive from the molecular dynamics of specific ion channels in the cell membrane. By analogy with other non-linear wave problems, we thus understand that these signals have propagated over a long distance — e.g. $\approx$ one meter from touch receptors in a finger to their targets in the spinal cord — so that every spike has the same shape. This is an important observation since it implies that all information carried by a spike train is encoded in the arrival times of the spikes. Since a creature's brain is connected to all of its sensory systems by such axons, all the creature knows about the outside world must be encoded in spike arrival times.

Until recently, neural codes have been studied primarily by measuring changes in the rate of spike production by different input signals. Recently it has become possible to characterize the codes in information-theoretic terms, and this has led to the discovery of some potentially universal features of the code (Bialek, 1996) (or see (Bialek, 1993) for a brief summary). They are:

1. *Very high information rates.* The record so far is 300 bits per second in a cricket mechanical sensor.

2. *High coding efficiency.* In cricket and frog vibration sensors, the information rate is within a factor of 2 of the entropy per unit time of the spike train.

3. *Linear decoding.* Despite evident non-linearities of the nervous system, spike trains can be *decoded* by simple linear filters. Thus we can write an estimate of the analog input signal $s(t)$ as $s_{\text{est}}(t) = \sum_i K_1(t - t_i)$, with $K_1$ chosen to minimize the mean-squared errors ($\chi^2$) in the estimate. Adding non-linear $K_2(t - t_i, t - t_j)$ terms does not significantly reduce $\chi^2$.

4. *Moderate signal-to-noise ratios (SNR).* The SNR in these experiments was defined as the ratio of power spectra of the input signal to the noise referred back to the input; the power spectrum of the noise was approximated by $\chi^2$ defined above. All these examples of high information transmission rates have SNR of order unity over a broad bandwidth, rather than high SNR in a narrow band.

We will try to tie all of these observations together by elevating the first to a principle: The neural code is chosen to maximize information transmission where information is quantified following Shannon. We apply this principle in the context of a simple model neuron which converts analog signals into spike trains. Before we consider a specific model, we will present a procedure for expanding the information rate of any point process encoding of an analog signal about the limit where the spikes are uncorrelated. We will briefly discuss how this can be used to measure information rates in real neurons.

This work will also appear in Network.

## 2   Information Theory

In the 1940's, Shannon proposed a quantitative definition for "information" (Shannon, 1949). He argued first that the average amount of information gained by observing some event $x$ is the entropy of the distribution from which $x$ is chosen, and then showed that this is the only definition consistent with several plausible requirements. This definition implies that the amount of information one signal can provide about some other signal is the difference between the entropy of the first signal's *a priori* distribution and the entropy of its conditional distribution. The average of this quantity is called the mutual (or transmitted) information. Thus, we can write the amount of information that the spike train, $\{t_i\}$, tells us about the time dependent signal, $s(t)$, as

$$I = -\int \frac{\mathcal{D}t_i}{N!} P[\{t_i\}] \log_2 P[\{t_i\}] - \left\langle -\int \frac{\mathcal{D}t_i}{N!} P[\{t_i\}|s()] \log_2 P[\{t_i\}|s()] \right\rangle_s, \quad (1)$$

where $\int \mathcal{D}t_i$ is shorthand for integration over all arrival times $\{t_i\}$ from 0 to $T$ and summation over the total number of spikes, $N$ (we have divided the integration measure by $N!$ to prevent over counting due to equivalent permutations of the spikes, rather than absorb this factor into the probability distribution as we did in (DeWeese [1], 1995)). $< \cdots >_s \equiv \int \mathcal{D}s P[s()] \cdots$ denotes integration over the space of functions $s(t)$ weighted by the signal's *a priori* distribution, $P[\{t_i\}|s()]$ is the probability distribution for the spike train when the signal is fixed and $P[\{t_i\}]$ is the spike train's average distribution.

## 3   Arbitrary Point Process Encoding of an Analog Signal

In order to derive a useful expression for the information given by Eq. (1), we need an explicit representation for the conditional distribution of the spike train. If we choose to represent each spike as a Dirac delta function, then the spike train can be defined as

$$\rho(t) \equiv \sum_{i=1}^{N} \delta(t - t_i). \quad (2)$$

This is the output spike train for our cell, so it must be a functional of both the input signal, $s(t)$, and all the noise sources in the cell which we will lump together and call $\eta(t)$. Choosing to represent the spikes as delta functions allows us to think of $\rho(t)$ as the probability of finding a spike at time $t$ when both the signal and noise are specified. In other words, if the noise were not present, $\rho$ would be the cell's firing rate, singular though it is. This implies that in the presence of noise the cell's observed firing rate, $r(t)$, is the noise average of $\rho(t)$:

$$r(t) = \int \mathcal{D}\eta P[\eta()|s()]\rho(t) \equiv \langle \rho(t) \rangle_\eta. \quad (3)$$

Notice that by averaging over the conditional distribution for the noise rather than its *a priori* distribution as we did in (DeWeese [1], 1995), we ensure that this expression is still valid if the noise is signal dependent, as is the case in many real neurons.

For any particular realization of the noise, the spike train is completely specified which means that the distribution for the spike train when both the signal and

noise are fixed is a modulated Poisson process with a singular firing rate, $\rho(t)$. We emphasize that this is true even though we have assumed nothing about the encoding of the signal in the spike train when the noise is not fixed. One might then assume that the conditional distribution for the spike train for fixed signal would be the noise average of the familiar formula for a modulated Poisson process:

$$P[\{t_i\}|s()] \approx \left\langle e^{-\int_o^T dt\rho(t)} \prod_{i=1}^N \rho(t_i) \right\rangle_\eta . \qquad (4)$$

However, this is only approximately true due to subtleties arising from the singular nature of $\rho(t)$. One can derive the correct expression (DeWeese [1], 1995) by carefully taking the continuum limit of an approximation to this distribution defined for discrete time. The result is the same sum of noise averages over products of $\rho$'s produced by expanding the exponential in Eq. (4) in powers of $\int dt\rho(t)$ *except* that all terms containing more than one factor of $\rho(t)$ at equal times are not present. The exact answer is:

$$P[\{t_i\}|s()] = \left\langle e^{-\int dt\rho(t)} \prod_{i=1}^N \rho(t_i) \right\rangle_\eta^- , \qquad (5)$$

where the superscripted minus sign reminds us to remove all terms containing products of coincident $\rho$'s after expanding everything in the noise average in powers of $\rho$.

## 4   Expanding About the Poisson Limit

An exact solution for the mutual information between the input signal and spike train would be hopeless for all but a few coding schemes. However, the success of linear decoding coupled with the high information rates seen in the experiments suggests to us that the spikes might be transmitting roughly independent information (see (DeWeese [1], 1995) or (Bialek, 1993) for a more fleshed out argument on this point). If this is the case, then the spike train should approximate a Poisson process. We can explicitly show this relationship by performing a cluster expansion on the right hand side of Eq. (5):

$$
\begin{aligned}
P[\{t_i\}|s()] &= e^{-\int r(t)dt} \prod_{i=1}^N r(t_i) \left\langle e^{-\int \Delta\rho(t)dt} \prod_{i=1}^N \left(1 + \frac{\Delta\rho(t_i)}{r(t_i)}\right) \right\rangle_\eta^- \qquad (6) \\
&= e^{-\int r(t)dt} \prod_{i=1}^N r(t_i) \left[1 + \sum_{m=2}^\infty C_\eta(m)\right],
\end{aligned}
$$

where we have defined $\Delta\rho(t) \equiv \rho(t) - <\rho(t)>_\eta = \rho(t) - r(t)$ and introduced $C_\eta(m)$ which collects all terms containing $m$ factors of $\Delta\rho$. For example,

$$C_\eta(2) \equiv \frac{1}{2}\sum_{i \neq j} \frac{\langle \Delta\rho_i \Delta\rho_j \rangle_\eta^-}{r_i r_j} - \int dt' \sum_{i=1}^N \frac{\langle \Delta\rho' \Delta\rho_i \rangle_\eta^-}{r_i} + \frac{1}{2}\int dt' dt'' \langle \Delta\rho' \Delta\rho'' \rangle_\eta^- . \quad (7)$$

Clearly, if the correlations between spikes are small in the noise distribution, then the $C_\eta$'s will be small, and the spike train will nearly approximate a modulated Poisson process when the signal is fixed.

Performing the cluster expansion on the signal average of Eq. (5) yields a similar expression for the average distribution for the spike train:

$$P[\{t_i\}] = e^{-T\bar{r}}\bar{r}^N \left[1 + \sum_{m=2}^{\infty} C_{\eta,s}(m)\right], \qquad (8)$$

where $T$ is the total duration of the spike train, $\bar{r}$ is the average firing rate, and $C_{\eta,s}(m)$ is identical to $C_\eta(m)$ with these substitutions: $r(t) \to \bar{r}$, $\Delta\rho(t) \to \bar{\Delta}\rho(t) \equiv \rho(t) - \bar{r}$, and $\langle\cdots\rangle_\eta^- \to \langle\langle\cdots\rangle_\eta^-\rangle_s$. In this case, the distribution for a homogeneous Poisson process appears in front of the square brackets, and inside we have $1 +$ corrections due to correlations in the average spike train.

## 5   The Transmitted Information

Inserting these expressions for $P[\{t_i\}|s()]$ and $P[\{t_i\}]$ (taken to *all* orders in $\Delta\rho$ and $\bar{\Delta}\rho$, respectively) into Eq. (1), and expanding to second non-vanishing order in $\bar{r}\tau_c$ results in a useful expression for the information (DeWeese [1], 1995):

$$\begin{aligned} I = & \int_0^T dt \left\langle r(t) \log_2\left(\frac{r(t)}{\bar{r}(t)}\right)\right\rangle_s \\ & + \frac{1}{2}\int_0^T dt \int_0^T dt' \left[\left\langle \langle\rho\rho'\rangle_\eta^- \log_2\left(\frac{\bar{r}\bar{r}'\langle\rho\rho'\rangle_\eta^-}{rr'\langle\langle\rho\rho'\rangle_\eta^-\rangle_s}\right)\right\rangle_s \right. \\ & \left. - \frac{\langle\langle\Delta\rho\Delta\rho'\rangle_\eta^-\rangle_s}{\ln 2} + \frac{\langle\langle\bar{\Delta}\rho\bar{\Delta}\rho'\rangle_\eta^-\rangle_s}{\ln 2}\right] + \mathcal{O}\left[(\bar{r}\tau_c)^2\right]. \end{aligned} \qquad (9)$$

where we have suppressed the explicit time notation in the correction term inside the double integral. If the signal and noise are stationary then we can replace the $\int_0^T dt$ in front of each of these terms by $T$ illustrating that the information does indeed grow linearly with the duration of the spike train.

The leading term, which is exact if there are no correlations between the spikes, depends only on the firing rate, and is never negative. The first correction is positive when the correlations between pairs of spikes are being used to encode the signal, and negative when individual spikes carry redundant information. This correction term is cumbersome but we present it here because it is experimentally accessible, as we now describe.

This formula can be used to measure information rates in real neurons without having to assume any method of reconstructing the signal from the spike train. In the experimental context, averages over the (conditional) noise distribution become repeated trials with the same input signal, and averages over the signal are accomplished by summing over all trials. $r(t)$, for example, is the histogram of the spike trains resulting from the same input signal, while $\bar{r}(t)$ is the histogram of all spike trains resulting from all input signals. If the signal and noise are stationary, then $\bar{r}$ will not be time dependent. $\langle\rho(t)\rho(t')\rangle_\eta$ is in general a 2-dimensional histogram which is signal dependent: It is equal to the number of spike trains resulting from some specific input signal which simultaneously contain a spike in the time bins containing $t$ and $t'$. If the noise is stationary, then this is a function of only $t - t'$, and it reduces to a 1-dimensional histogram.

In order to measure the full amount of information contained in the spike train, it is crucial to bin the data in small enough time bins to resolve all of the structure in

$r(t)$, $\langle \rho(t)\rho(t')\rangle_\eta$, and so on. We have assumed nothing about the noise or signal; in fact, they can even be correlated so that the noise averages are signal dependent without changing the experimental procedure. The experimenter can also choose to fix only some aspects of the sensory data during the noise averaging step, thus measuring the mutual information between the spike train and only these aspects of the input. The only assumption we have made up to this point is that the spikes are roughly uncorrelated which can be checked by comparing the leading term to the first correction, just as we do for the model we discuss in the next section.

## 6  The Linear Filtered Threshold Crossing Model

As we reported in a previous proceedings (DeWeese [2], 1995) (and see (DeWeese [1], 1995) for details), the leading term in Eq. (9) can be calculated exactly in the case of a linear filtered threshold crossing (LFTC) model when the signal and noise are drawn from independent Gaussian distributions. Unlike the Integrate and Fire (IF) model, the LFTC model does not have a "renewal process" which resets the value of the filtered signal to zero each time the threshold is reached. Stevens and Zador have developed an alternative formulation for the information transmission which is better suited for studying the IF model under some circumstances (Stevens, 1995), and they give a nice discussion on the way in which these two formulations compliment each other.

For the LFTC model, the leading term is a function of only three variables: 1) The threshold height; 2) the ratio of the variances of the filtered signal and the filtered noise, $\langle s^2(t)\rangle_s / \langle \eta^2(t)\rangle_\eta$, which we refer to as the SNR; 3) and the ratio of correlation times of the filtered signal and the filtered noise, $\tau_s/\tau_\eta$, where $\tau_s^2 \equiv \langle s^2(t)\rangle_s / \langle \dot{s}^2(t)\rangle_s$ and similarly for the noise. In the equations in this last sentence, and in what follows, we absorb the linear filter into our definitions for the power spectra of the signal and noise. Near the Poisson limit, the linear filter can only affect the information rate through its generally weak influence on the ratios of variances and correlation times of the signal and noise, so we focus on the threshold to understand adaptation in our model cell.

When the ratio of correlation times of the signal and noise is moderate, we find a maximum for the information rate near the Poisson limit — the leading term $\approx 10\times$ the first correction. For the interesting and physically relevant case where the noise is slightly more broadband than the signal as seen through the cell's prefiltering, we find that the maximum information rate is achieved with a threshold setting which does not correspond to the maximum average firing rate illustrating that this optimum is non-trivial. Provided the SNR is about one or less, linear decoding does well — a lower bound on the information rate based on optimal linear reconstruction of the signal is within a factor of two of the total available information in the spike train. As SNR grows unbounded, this lower bound asymptotes to a constant. In addition, the required timing resolution for extracting the information from the spike train is quite modest — discretizing the spike train into bins which are half as wide as the correlation time of the signal degrades the information rate by less than 10%. However, at maximum information transmission, the information *per spike* is low — $R_{max}/\bar{r} \approx .7$ bits/spike, much lower than 3 bits/spike seen in the cricket. This low information rate drives the efficiency down to 1/3 of the experimental values despite the model's robustness to timing jitter. Aside from the low information rate, the optimized model captures all the experimental features we set out to explain.

# 7  Concluding Remarks

We have derived a useful expression for the transmitted information which can be used to measure information rates in real neurons provided the correlations between spikes are shorter range than the average inter-spike interval. We have described a method for checking this hypothesis experimentally. The four seemingly unrelated features that were common to several experiments on a variety of neurons are actually the natural consequences of maximizing the transmitted information. Specifically, they are all due to the relation between $\bar{r}$ and $\tau_c$ that is imposed by the optimization. We reiterate our previous prediction (DeWeese [2], 1995; Bialek, 1993): Optimizing the code requires that the threshold adapt not only to cancel DC offsets, but it must adapt to the statistical structure of the signal and noise. Experimental hints at adaptation to statistical structure have recently been seen in the fly visual system (de Ruyter van Steveninck, 1994) and in the salamander retina (Warland, 1995).

# 8  References

M. DeWeese 1995 *Optimization Principles for the Neural Code* (Dissertation, Princeton University)

M. DeWeese and W. Bialek 1995 Information flow in sensory neurons *Il Nuovo Cimento* **17D** 733-738

E. D. Adrian 1928 *The Basis of Sensation* (New York: W. W. Norton)

F. Rieke, D. Warland, R. de Ruyter van Steveninck, and W. Bialek 1996 *Neural Coding* (Boston: MIT Press)

W. Bialek, M. DeWeese, F. Rieke, and D. Warland 1993 Bits and Brains: Information Flow in the Nervous System *Physica A* **200** 581-593

C. E. Shannon 1949 Communication in the presence of noise, *Proc. I. R. E.* **37** 10-21

C. Stevens and A. Zador 1996 *Information Flow Through a Spiking Neuron* in M. Hasselmo ed *Advances in Neural Information Processing Systems, Vol 8* (Boston: MIT Press) (this volume)

R.R. de Ruyter van Steveninck, W. Bialek, M. Potters, R.H. Carlson 1994 Statistical adaptation and optimal estimation in movement computation by the blowfly visual system, in *IEEE International Conference On Systems, Man, and Cybernetics* pp 302-307

D. Warland, M. Berry, S. Smirnakis, and M. Meister 1995 personal communication
